# Similarity and discrimination in classical conditioning: A latent variable account

**Aaron C. Courville*[1,3], Nathaniel D. Daw[4] and David S. Touretzky[2,3]**
[1]Robotics Institute, [2]Computer Science Department,
[3]Center for the Neural Basis of Cognition,
Carnegie Mellon University, Pittsburgh, PA 15213
[4]Gatsby Computational Neuroscience Unit, University College London
{aaronc,dst}@cs.cmu.edu; daw@gatsby.ucl.ac.uk

## Abstract

We propose a probabilistic, generative account of configural learning phenomena in classical conditioning. Configural learning experiments probe how animals discriminate and generalize between patterns of simultaneously presented stimuli (such as tones and lights) that are differentially predictive of reinforcement. Previous models of these issues have been successful more on a phenomenological than an explanatory level: they reproduce experimental findings but, lacking formal foundations, provide scant basis for understanding why animals behave as they do. We present a theory that clarifies seemingly arbitrary aspects of previous models while also capturing a broader set of data. Key patterns of data, e.g. concerning animals' readiness to distinguish patterns with varying degrees of overlap, are shown to follow from statistical inference.

## 1   Introduction

Classical conditioning experiments probe how organisms learn to predict significant events such as the receipt of food or shock. While there is a history of detailed quantitative theories about these experiments, only recently has there been a sustained attempt to understand them in terms of sound statistical prediction [1]. A statistical foundation helps to identify key theoretical issues (such as uncertainty) underlying these experiments, to explain otherwise puzzling results, and to connect these behavioral theories with theories of neural computation, which are also increasingly framed in statistical terms.

A cluster of issues that has received great experimental and theoretical attention in conditioning — but not yet from a statistically grounded perspective — concerns discrimination and generalization between patterns of sensory input. Historically, these issues arose in the context of nonlinear discriminations, such as the XOR problem (in which, e.g., a light and a tone each predict shock when presented alone, but not together). While animals can learn such a discrimination, the seminal model of Rescorla and Wagner [2] cannot, since it assumes that the prediction is linear in the stimuli. Traditionally, this problem was solved by introducing extra discriminative features to the model's input (known as "configural units," since they detect conjunctions of stimuli such as tone plus light), rendering the augmented problem linearly solvable [3]. On this foundation rests a wealth of work probing how ani-

mals learn and predict given compounds of stimuli. Here, we reinterpret these issues from a Bayesian perspective.

Previous work posits an informal division (or perhaps a spectrum) between "elemental" and "configural" approaches to stimulus patterns, distinguished by whether a compound's association with reinforcement is derived from its individual stimuli (lights, tones), or rests collectively in the full compound (light and tone together). The prototypical elemental model is the original Rescorla–Wagner model, without configural units, in which the aggregate prediction is linear in the elements. The standard configural model is that of Pearce [4], in which responding to a compound is determined by previous experience with that and other similar compounds, through a process of generalization and weighted averaging. Both theories match an impressive range of experimental data, but each is refuted by some experiments that the other captures. It is not clear how to move beyond this stalemate. Because the theories lack formal foundations, their details — particularly those on which they differ — are ad-hoc and poorly understood. For instance, what circumstances justify the introduction of a new configural unit, and what should be the form of generalization between compounds?

Here we leverage our Bayesian theory of conditioning [5] to shed new light on these issues. Our model differs from traditional ones in a number of ways. Notably, analogizing conditioning to classification, we take a generative rather than a discriminative approach. That is, we assume animals are modeling their complete sensory experience (lights, tones, and shocks) rather than only the chance of shock conditioned on lights and tones. We assume that stimuli are correlated with each other, and with reinforcement, through shared latent variables. Because a latent variable can trigger multiple events, these causes play a role akin to configural units in previous theories, but offer stronger normative guidance. Questions about generalization (what is the probability that a latent variable is active given a particular constellation of inputs) are seen as standard statistical inference; questions about model structure (how many "configural units" should there be and with what constellations of stimuli are they associated) are answerable using Bayesian model averaging, which we have suggested animals can approximate [5]. Such inferences also determine whether an animal's experience on a trial is best explained by multiple causes interacting additively, in the style of Rescorla–Wagner, or by a single cause triggering multiple events like one of Pearce's configural units. This allows our theory to capture patterns of data that seem to favor each of its predecessors.

Our theory is meant to shed light on the normative reasons why animals behave as they do, rather than on how they might carry out computations like those we describe. In practice, the inferences we discuss can be computed only approximately, and we intend no claim that animals are using the same approximations to them as we are. More mechanistic models, such as Pearce's, can broadly be viewed as plausible implementations for approximating some aspects of our more general framework.

## 2 Theories of Learning with Compound Stimuli

Classical conditioning experiments probe animals' anticipation of a reinforcer $R$ such as food or footshock, given the presentation of initially neutral stimuli such as lights and tones. Expectation is assessed via reflexive conditioned responses such as salivation or freezing, which are thought to reveal animals' predictions of reinforcement. By studying responding as a function of the pattern of previous reinforcer / stimulus pairings, the experiments assess learning. To describe a conditioning task abstractly, we use capital letters for the stimuli and $+$ and $-$ to indicate whether they are reinforced. For instance, the XOR task can be written as $A+$, $B+$, $AB-$, where $AB-$ denotes simultaneous presentation of both stimuli unreinforced. Typically, each type of trial is delivered repeatedly, and the development of responding is assessed.

We now describe the treatment of compound stimuli in the models of Rescorla and Wagner [2] and Pearce [4]. In both models, the set of stimuli present on a trial is converted into an input vector $\mathbf{x}$. The strength of the conditioned response is modeled as proportional to a prediction of reinforcement $v = \mathbf{x} \cdot \mathbf{w}$, the dot product between the input and a weight vector. Finally, one or more weights are updated proportionally to the mismatch $r - v$ between observed and predicted reinforcement.

For both theories, $\mathbf{x}$ includes an element (or "unit") corresponding to each individual stimulus.[1] In Pearce's model, and in augmented "added elements" versions of the Rescorla–Wagner model [3], additional "configural" units are also included, corresponding to conjunctions of stimuli. In particular, it is assumed that a unique configural unit is added for each stimulus compound observed, such as $ABC$. Note that this assumption is both arbitrary (e.g. we might very well include elements for subcompounds such as $AB$) and unrealistic (given the profusion of uncontrolled stimuli simultaneously present in a real experiment).

The theories differ as to how they apportion activation over $\mathbf{x}$ and learning over $\mathbf{w}$. In the Rescorla–Wagner model, the input vector is binary: $x_i = 1$ if the $i$th stimulus (or an exactly matching compound) is present, $0$ otherwise. For learning, the weight corresponding to each active input is updated. The Pearce model instead spreads graded activation over $\mathbf{x}$, based on a measure of similarity between the observed stimulus compound (or element) and the compounds represented by the model's configural units. In particular, if we denote the number of stimulus elements present in an observed stimulus pattern $a$ as $\mathrm{size}(a)$, and in the pattern represented by the $i$th configural unit as $\mathrm{size}(i)$, then the activation of unit $i$ by pattern $a$ is given by $x_i = \mathrm{size}(\mathrm{overlap}(a,i))^2/(\mathrm{size}(a) \cdot \mathrm{size}(i))$. The learning phase updates only the weight corresponding to the configural unit that exactly matches the observed stimulus configuration.

As neither scheme has much formal basis, there seems to be no *theoretical* reason to prefer one over the other, nor over any other ad-hoc recipe for apportioning representation and learning. *Empirical* considerations also provide ambivalent guidance, as we discuss next.

## 3   Data on Learning with Compound Stimuli

Both the elemental and configural models reproduce a number of well known experimental phenomena. Here we review several basic patterns of results. Notably, each theory has a set of experiments that seems to support it over the other. Later, we will show that our normative theory accounts for all of these results.

**Overshadowing**   When a pair of stimuli $AB+$ is reinforced together, then tested separately, responding to either individual stimulus is often attenuated compared to a control in which the stimulus is trained alone ($A+$). Both models reproduce overshadowing, though Rescorla–Wagner incorrectly predicts that it takes at least two $AB+$ pairings to materialize.

**Summation**   The converse of overshadowing is summation: when two stimuli are individually reinforced, then tested together, there is often a greater response to the pair than to either element alone. In a recent variation by Rescorla [6], animals were trained on a pair of compounds $AB+$ and $CD+$, then responses were measured to the trained compounds, the individual elements $A$, $B$, etc., and the novel transfer compounds $AD$ and $BC$. The strongest response was elicited by the trained compounds. The transfer compounds elicited a moderate response, and the individual stimuli produced the weakest responding.

The added elements Rescorla–Wagner model predicts this result due to the linear summation of the influences of all the units ($A$ through $D$, $AB$, and $CD$ — note that the added configural units are crucial). However, because of the normalization term in the generalization rule, Pearce's model often predicts no summation. Here it predicts equal responding to the individual stimuli and to the transfer compounds. There is controversy as to whether the model can realistically be reconciled with summation effects [4, 7], but on the whole, these phenomena seem more parsimoniously explained with an elemental account.

**Overlap**   A large number of experiments (see [4] for a review) demonstrate that the more elements shared by two compounds, the longer it takes animals to learn to discriminate between them. Though this may seem intuitive, elemental theories predict the opposite. In one example, Redhead and Pearce [8] presented subjects with the patterns $A+$, $BC+$ reinforced and $ABC-$ unreinforced. Differential responding between $A$ and $ABC$ was achieved in fewer trials than that between $BC$ and $ABC$.

Pearce's configural theory predicts this result because the extra overlap between $BC$ and $ABC$ (compared to $A$ vs. $ABC$) causes each compound to activate the other's configural unit more strongly. Thus, larger weights are required to produce a differentiated prediction. Rescorla–Wagner predicts the opposite result, because compounds with more elements, e.g. $BC$, accumulate more learning on each trial.

# 4   A latent variable model of stimulus generalization

In this section we present a *generative model* of how stimuli and reinforcers are jointly delivered. We will show how the model may be used to estimate the conditional probability of reinforcement (the quantity we assume drives animals' responding) given some pattern of observed stimuli. The theory is based on the one we presented in [5], and casts conditioning as inference over a set of sigmoid belief networks. Our goal here is to use this formalism to explain configural learning phenomena.

## 4.1   A Sigmoid Belief Network Model of Conditioning

Consider a vector of random variables $\mathbf{S}$ representing stimuli on a trial, with the $j$th stimulus present when $S_j = 1$ and absent when $S_j = 0$. One element of $\mathbf{S}$ is distinguished as the reinforcer $R$; the remainder (lights and tones) is denoted as $Stim$. We encode the correlations between all stimuli (including the reinforcer) through common connections to a vector of latent variables, or *causes*, $\mathbf{x}$ where $x_i \in \{0, 1\}$. According to the generative process, on each trial the state of the latent variables is determined by independent Bernoulli draws (each latent variable has a weight determining its chance of activation [5]). The probability of stimulus $j$ being present is then determined by its relationship to the latent variables:

$$P(S_j \mid m, \mathbf{w}_m, \mathbf{x}) = (1 + \exp(-(\mathbf{w}_m^{(j)})^T \mathbf{x} - w_{bias}))^{-1}, \tag{1}$$

where the weight vector $\mathbf{w}_m^{(j)}$ encodes the connection strengths between $\mathbf{x}$ and $S_j$ for the model structure $m$. The bias weight $w_{bias}$ is fixed at $-6$, ensuring that spontaneous events are rare. Some examples of the type of network structure under consideration are shown as graphical models in Figure 1(c)–(d) and Figure 2(c)–(e).

We assume animals learn about the model structure itself, analogous to the experience-dependent introduction of configural units in previous theories. In our theory, animals use experience to infer which network structures (from a set of candidates) and weights likely produced the observed stimuli and reinforcers. These in turn determine predictions of future reinforcement. Details of this inference are laid out below.

## 4.2 Generalization: inference over latent variables

Generalization between observed stimulus patterns is a key aspect of previous models. We now describe how generalization arises in our theory.

Given a particular belief net structure $m$, weights $\mathbf{w}_m$, and previous conditioning experience $\mathcal{D}$, the probability of reinforcement $R$ given observed stimuli $Stim$ can be computed by integrating over the possible settings $\mathbf{x}$ of the latent variables:

$$P(R \mid Stim, m, \mathbf{w}_m, \mathcal{D}) = \sum_{\mathbf{x}} P(R \mid m, \mathbf{w}_m, \mathbf{x}) P(\mathbf{x} \mid Stim, m, \mathbf{w}_m, \mathcal{D}) \qquad (2)$$

The first term is given by Equation 1. By Bayes' rule, the second term weighs particular settings of the hidden causes proportionally to the likelihood that they would give rise to the observed stimuli. This process is a counterpart to Pearce's generalization rule for configural units. Unlike Pearce's rule, inference over $\mathbf{x}$ considers settings of the individual causes $x_i$ jointly (allowing for *explaining away* effects) and incorporates prior probabilities over each cause's activation. Nevertheless, the new rule broadly resembles its predecessor in that a cause is judged likely to be active (and contributes to predicting $R$) if the constellation of stimuli it predicts is similar to what is observed.

## 4.3 Learning to discriminate: inference over models

We treat the model weights $\mathbf{w}_m$ and the model structure $m$ as uncertain quantities subject to standard Bayesian inference. We assume that, given a model structure, the weights are mutually independent *a priori* and each distributed according to a Laplace distribution.[2] Conditioning on the data $\mathcal{D}$ produces a posterior distribution over the weights, over which we integrate to predict $R$:

$$P(R \mid Stim, m, \mathcal{D}) = \int P(R \mid Stim, m, \mathbf{w}_m, \mathcal{D}) P(\mathbf{w}_m \mid m, \mathcal{D}) d\mathbf{w}_m \qquad (3)$$

Uncertainty over model structure is handled analogously. Integrating over posterior model uncertainty we arrive at the prediction of reinforcement:

$$P(R \mid Stim, \mathcal{D}) = \sum_m P(R \mid Stim, m, \mathcal{D}) P(m \mid \mathcal{D}), \qquad (4)$$

where $P(m \mid \mathcal{D}) \propto P(\mathcal{D} \mid m) P(m)$ and the marginal likelihood $P(\mathcal{D} \mid m)$ is computed similarly to equation 3, by integration over the weights. The prior over models, $P(m)$ is expressed as a distribution over $n_{\mathbf{x}}$, the number of latent variables, and over $l_i$, the number of links between the stimuli and each latent variable: $P(m) = P(n_{\mathbf{x}}) \prod_{i=1}^{n_{\mathbf{x}}} P(l_i)$. We assume that $P(n_{\mathbf{x}})$ and each $P(l_i)$ are given by geometric distributions (param. = 0.1), renormalized to sum to unity over the maximum of 5 latents and 5 stimuli. This prior reflects a bias against complex models. The marginal likelihood term also favors simplicity, due to the *automatic Occam's razor* (see [5]). For our simulations, we approximately evaluated Equation 4 using reversible-jump Markov Chain Monte Carlo (see [5] for details).

Progressively conditioning on experience to resolve prior uncertainty in the weights and model structure produces a gradual change in predictions akin to the incremental learning rules of previous models. The extent to which a particular model structure $m$ participates in predicting $R$ in Equation 4 is, by Bayes' rule, proportional to its prior probability, $P(m)$, and to the extent that it explains the data, $P(\mathcal{D} \mid m)$. Thus a prior preference for simpler models competes against better data fidelity for more complex models. As data accumulate,

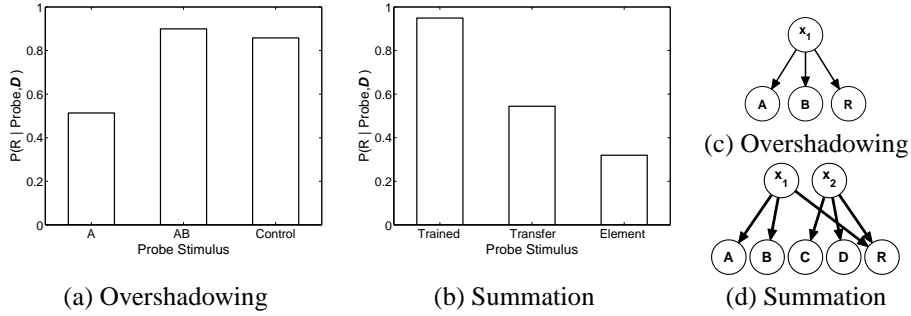

(a) Overshadowing      (b) Summation      (d) Summation

Figure 1: Results of MCMC simulation. (a) Overshadowing ($AB+$): the predicted probability of reinforcement in response to presentations of the element $A$, the compound $AB$, and an individually trained control element ($A+$). (b) Summation experiment ($AB+$, $CD+$): the predicted probability of reinforcement in response to separate presentations of the trained compounds ($AB$, $CD$), the transfer compounds ($AD$, $BC$) and the elements ($A$, $B$, etc.). (c) Depiction of the MAP model structure after overshadowing training. (d) The MAP model structure after $AB+$ $CD+$ training.

the balance shifts toward the latter, and predictions become more accurate. Analogously, weights are small *a priori* but can grow with experience.

Together with the generalization effects discussed above, these inference effects explain why animals can learn more readily to discriminate stimulus compounds that have less overlap. Key to the discrimination is inferring that different compounds are produced by separate latent variables; the more the compounds overlap, the more accurately will the data be approximated by a model with a single latent variable (preferred *a priori*), which biases the complexity-fidelity tradeoff toward simplicity and retards acquisition.

## 5    Results

**Overshadowing**    Overshadowing exemplifies our account of between-compound generalization; our model's performance is illustrated in Figure 1(a). After 5 $AB+$ pairings, the network with highest posterior probability, depicted in (c), contains one latent variable correlated with both stimuli and the reinforcer. Consistent with experimental results, testing on $A$ produces attenuated responding. This is because predicting whether $A$ is reinforced requires balancing the relative probabilities of two unlikely events: that the stimulus occurred spontaneously (with $x_1$ inactive), versus that it was caused by $x_1$ being active, but that $B$ uncharacteristically failed to occur (this probability measures *generalization* between the patterns $A$ and $AB$). Overall, this tradeoff decreases the chance that $x_1$ is active, suppressing the prediction of reinforcement relative to the control treatment, where $A$ is reinforced in isolation ($A+$). Unlike the Rescorla–Wagner model, ours correctly predicts that overshadowing can occur after even a single $AB+$ presentation.

**Summation**    Figure 1(b) shows our model's performance on Rescorla's $AB+$ $CD+$ summation and transfer experiment [6], which is one of several summation experiments our model explains. Compounds were reinforced 10 times. Consistent with experimental findings, the model predicts greatest responding to the trained compounds ($AB$, $CD$), moderate responding to transfer compounds ($AD$, $BC$), and least responding to the elements ($A$, $B$, etc.). The maximum *a posteriori* (MAP) model structure (Figure 1(d)) mimics the training compounds, with one latent variable connected to $A$, $B$, and $R$ and another connected to $C$, $D$, and $R$. The results follow from a combination of generalization and additivity. The training compounds activate one latent variable strongly; the transfer compounds acti-

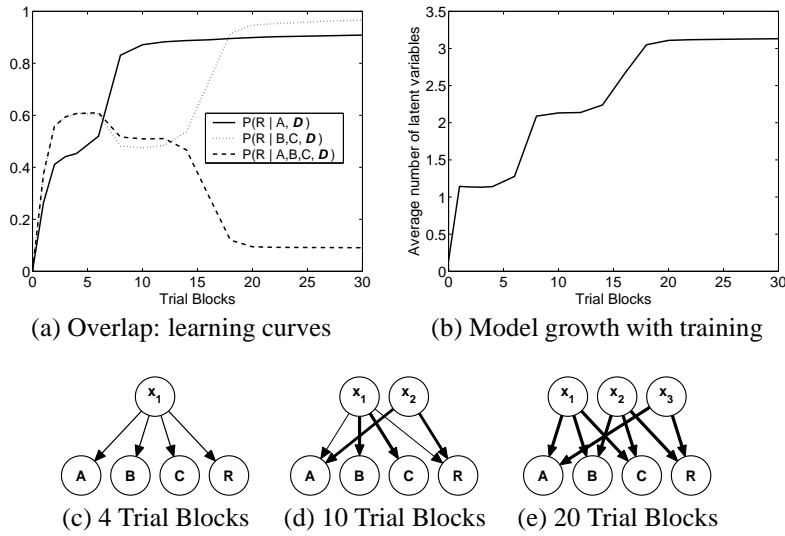

(a) Overlap: learning curves  (b) Model growth with training

(c) 4 Trial Blocks  (d) 10 Trial Blocks  (e) 20 Trial Blocks

Figure 2: Summary of MCMC simulation results on the $A+$, $BC+$, $ABC-$ experiment. The estimated error due to MCMC sampling is small and not shown. (a) Learning curves showing the predicted probability of reinforcement in response to separate presentations of $A$, $BC$, and $ABC$ as a function of number of trial blocks. (b) The average number of latent variables over the 10000 MCMC sample models. (c) - (e) Representations of MAP model structures after training with 4, 10, and 20 trial blocks (edge widths represent mean weight strength).

vate both latents weakly (together additively influencing the probability of reinforcement); the elements weakly activate only a single latent variable.

**Overlap**  Figure 2(a) shows the model's learning curves from the overlapping compound experiment, $A+$, $BC+$, $ABC-$. Each trial block contains one trial of each type. The model correctly predicts faster discrimination between $A$ and $ABC$ than between $BC$ and $ABC$. This pattern results from progressive increase in the number of inferred latent variables (b). Early in training, probability density concentrates on small models with a single latent variable correlating all stimuli and the reinforcer (c). After more trials, models with two latent variables become more probable, one correlating $A$ and $R$ and the other correlating $B$ and $C$ with both $A$ and $R$, attempting to capture both $BC+$ and $ABC-$ trial types. (d). With further training, the most likely models are those with three latents, each encoding one trial type (e). Our theory captures many similar experiments demonstrating the difficulty of discriminating overlapping compounds.

## 6   Discussion

The configural unit is an ad-hoc device that nonetheless plays a key role in previous experimental and theoretical work in conditioning. Its inclusion in models like that of Rescorla–Wagner invites a number of questions. Which configurations should be represented? How should activation and learning be apportioned between them? These issues are contentious, admitting no clear answer, precisely because of the arbitrary nature of the device. We have shown how a latent variable correlated with a constellation of stimuli provides a well founded counterpart to the configural unit, and how a range of experimental phenomena concerning similarity and discrimination can be accounted for with the assumption that animals are carrying out inference about these variables. While data exist that tend to fa-

vor each of the two major previous models of configural learning over the other, the new model accounts for the full pattern, balancing the strengths of both theories. Our theory also improves on its predecessors in other ways; for instance, because it includes learning about stimulus interrelationships it can explain second-order conditioning [5], which is not addressed by either the Pearce or the Rescorla–Wagner accounts.

Of course, many issues remain. A full account of summation phenomena, in particular, is beyond the scope of the present model. We treat reinforcer delivery as binary and model a limited, saturating, summation in probabilities. However, realistic summation almost certainly concerns reinforcement *magnitudes* as well (see, for example, [9]), and our model would need to be augmented to address them. Because we have assumed that trials are IID, the model cannot yet account for effects of trial ordering (e.g. the difference between partial reinforcement and extinction). These could be addressed by incorporating dynamics into the generative model, so that inference requires tracking the changing model parameters. Also for future work is exploring how different priors might give rise to different behavior. An advantage of Bayesian modeling is that because the free parameters are formulated as priors, they represent concrete assertions about the world (e.g. how often particular kinds of events occur), and can thus be constrained and even experimentally manipulated.

We have focused only on two previous models and only on animal behavioral experiments. Issues of similarity and discrimination are also studied in the rather different setting of human category judgments, where Bayesian generative approaches have also proved useful [10]. There is also a tradition of more neurophysiological models of the *hippocampal* substrates of configural learning [11, 12]. Given the large body of theory and experiment on these issues, this seems a promising direction for future work connecting our behavioral theory with neurophysiological ones. In one of the hippocampal theories, Gluck and Myers [12] augment the Rescorla–Wagner model with an input representation learned by an autoencoder. Since autoencoders perform probabilistic density modeling, this is probably the most statistically minded of prior approaches to configural representation and has clear parallels with our work.

### Acknowledgments

This work was supported by National Science Foundation grants IIS-9978403 and DGE-9987588. ND is funded by a Royal Society USA Research Fellowship and the Gatsby Foundation. We thank Peter Dayan, Yael Niv and Geoff Gordon for helpful discussions.

## Footnotes

[1]In Pearce's presentation of his model, these units are added only after elements are observed alone. We include them initially, which does not affect the model's behavior, to stress similarity with the Rescorla-Wagner model.

[2]The Laplace distribution is given by $f(y) = \frac{1}{2b} e^{-|y-\mu|/b}$. In our simulations $\mu = 0$ and $b = 2$. As a prior, it encodes a bias for sparsity consistent with a preference for simpler model structures.

## References

[1] P. Dayan, T. Long, *Advances in Neural Information Processing Systems 10* (1998), pp. 117–123.

[2] R. A. Rescorla, A. R. Wagner, *Classical Conditioning II*, A. H. Black, W. F. Prokasy, eds. (Appleton-Century-Crofts, 1972), pp. 64–99.

[3] R. A. Rescorla, *Journal of Comparative and Physiological Psychology* **79**, 307 (1972).

[4] J. M. Pearce, *Psychological Review* **101**, 587 (1994).

[5] A. C. Courville, N. D. Daw, G. J. Gordon, D. S. Touretzky, *Advances in Neural Information Processing Systems 16* (2004).

[6] R. A. Rescorla, *Quarterly Journal of Experimental Psychology* **56B**, 161 (2003).

[7] R. A. Rescorla, *Animal Learning and Behavior* **25**, 200 (1997).

[8] E. S. Redhead, J. M. Pearce, *Quarterly Journal of Experimental Psychology* **48B**, 46 (1995).

[9] E. F. Kremer, *Journal of Experimental Psychology: Animal Behavior Processes* **4**, 22 (1978).

[10] J. B. Tenenbaum, T. L. Griffiths, *Behavioral and Brain Sciences* **24**, 629 (2001).

[11] R. C. O'Reilly, J. W. Rudy, *Psychological Review* **108**, 311 (2001).

[12] M. A. Gluck, C. Myers, *Hippocampus* **3**, 491 (1993).
